# Nonlinear Pattern Separation in Single Hippocampal Neurons with Active Dendritic Membrane

Anthony M. Zador[†]     Brenda J. Claiborne[§]     Thomas H. Brown[†]

[†]Depts. of Psychology and Cellular
  & Molecular Physiology
Yale University
New Haven, CT 06511
zador@yale.edu

[§]Division of Life Sciences
University of Texas
San Antonio, TX 78285

## ABSTRACT

The dendritic trees of cortical pyramidal neurons seem ideally suited to perform local processing on inputs. To explore some of the implications of this complexity for the computational power of neurons, we simulated a realistic biophysical model of a hippocampal pyramidal cell in which a "cold spot"—a high density patch of inhibitory Ca-dependent K channels and a colocalized patch of Ca channels—was present at a dendritic branch point. The cold spot induced a nonmonotonic relationship between the strength of the synaptic input and the probability of neuronal firing. This effect could also be interpreted as an analog stochastic XOR.

## 1   INTRODUCTION

Cortical neurons consist of a highly branched dendritic tree that is electrically coupled to the soma. In a typical hippocampal pyramidal cell, over 10,000 excitatory synaptic inputs are distributed across the tree (Brown and Zador, 1990). Synaptic activity results in current flow through a transient conductance increase at the point of synaptic contact with the membrane. Since the primary means of rapid intraneuronal signalling is electrical, information flow can be characterized in terms of the electrical circuit defined by the synapses, the dendritic tree, and the soma.

Over a dozen nonlinear membrane channels have been described in hippocampal pyramidal neurons (Brown and Zador, 1990). There is experimental evidence for a heterogeneous distribution of some of these channels in the dendritic tree (*e.g.* Jones *et al.*, 1989). In the absence of these dendritic channels, the input-output function can sometimes be reasonably approximated by a modified sigmoidal model. Here we report that introducing a cold spot

at the junction of two dendritic branches can result in a fundamentally different, nonmonotonic input-output function.

## 2  MODEL

The biophysical details of the circuit class defined by dendritic trees have been well characterized (*reviewed in* Rall, 1977; Jack *et al.*, 1983). The fundamental circuit consists of a linear and a nonlinear component. The linear component can be approximated by a set of electrical compartments coupled in series (Fig. 1C), each consisting of a resistor and capacitor in parallel (Fig. 1B). The nonlinear component consists of a set of nonlinear resistors in parallel with the capacitance.

The model is summarized in Fig. 1A. Briefly, simulations were performed on a 3000-compartment anatomical reconstruction of a region CA1 hippocampal neuron (Claiborne *et al.*, 1992; Brown *et al.*, 1992). All dendritic membrane was passive, except at the cold spot (Fig. 1A). At the soma, fast K and Na channels (*cf.* Hodgkin-Huxley, 1952) generated action potentials in response to stimuli. The parameters for these channels were modified from Lytton and Sejnowski (1991; *cf.* Borg-Graham, 1991).

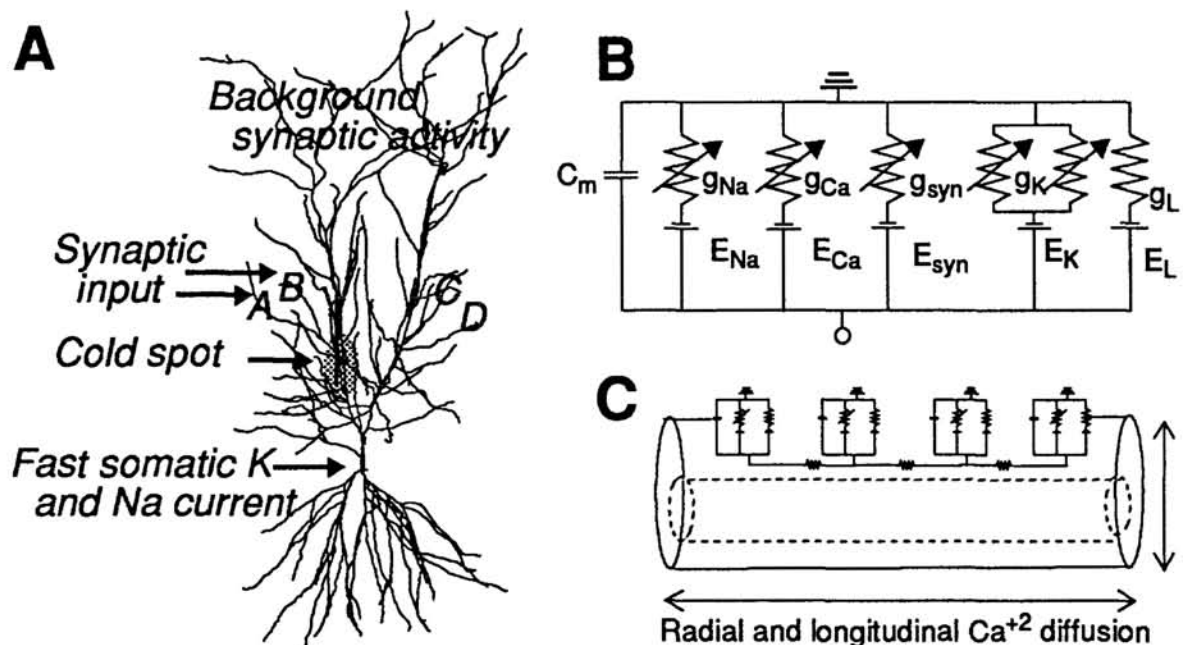

Fig. 1  The model. (A) The 3000-compartment electrical model used in these simulations was obtained from a 3-dimensional reconstruction of a hippocampal region CA1 pyramidal neuron (Claiborne et al, 1992). Each synaptic pathway (*A-D*) consisted of an adjustable number of synapses arrayed along the single branch indicated (*see text*). Random background activity was generated with a spatially uniform distribution of synapses firing according to Poisson statistics. The neuronal membrane was completely passive (linear), except at the indicated cold spot and at the soma. (B) In the nonlinear circuit associated with a patch a neuronal membrane containing active channels, each channel is described by a voltage-dependent conductance in series with its an ionic battery (*see text*). In the present model the channels were spatially localized, so no single patch contained all of the nonlinearities depicted in this hypothetical illustration. (C) A dendritic segment is illustrated in which both electrical and $Ca^{2+}$ dynamics were modelled. $Ca^{2+}$ buffering, and both radial and longitudinal $Ca^{2+}$ diffusion were simulated.

We distinguished four synaptic pathways *A-D* (*see* Fig. 1A). Each pathway consisted of a population of synapses activated synchronously. The synapses were of the fast *AMPA* type (*see* Brown *et. al.*, 1992). In addition, random background synaptic activity distributed uniformly across the dendritic tree fired according to Poisson statistics.

The cold spot consisted of a high density of a Ca-activated K channel, the *AHP* current (Lancaster and Nicoll, 1987; Lancaster *et. al.*, 1991) colocalized with a low density patch of N-type Ca channels (Lytton and Sejnowski, 1991; *cf.* Borg-Graham, 1991). Upon localized depolarization in the region of the cold spot, influx of $Ca^{2+}$ through the Ca channel resulted in a transient increase in the local $[Ca^{2+}]$. The model included $Ca^{2+}$ buffering, and both radial and longitudinal diffusion in the region of the cold spot. The increased $[Ca^{2+}]$ activated the inhibitory *AHP* current. The interplay between the direct excitatory effect of synaptic input, and its inhibitory effect via the *AHP* channels formed the functional basis of the cold spot.

## 3   RESULTS

### 3.1 DYNAMIC BEHAVIOR

Representative behavior of the model is illustrated in Fig. 2. The somatic potential is plotted as a function of time in a series of simulations in which the number of activated synapses in pathway *A/B* was increased from *0* to about *100*. For the first 100 *msec* of each simulation, background synaptic activity generated a noisy baseline. At $t = 100$ *msec*, the indicated number of synapses fired synchronously five times at *100 Hz*. Since the background activity was noisy, the outcome of the each simulation was a random process.

The key effect of the cold spot was to impose a limit on the maximum stimulus amplitude that caused firing, resulting in a window of stimulus strengths that triggered an action potential. In the absence of the cold spot a greater synaptic stimulus invariably increased the likelihood that a spike fired. This limit resulted from the relative magnitude of the *AHP*

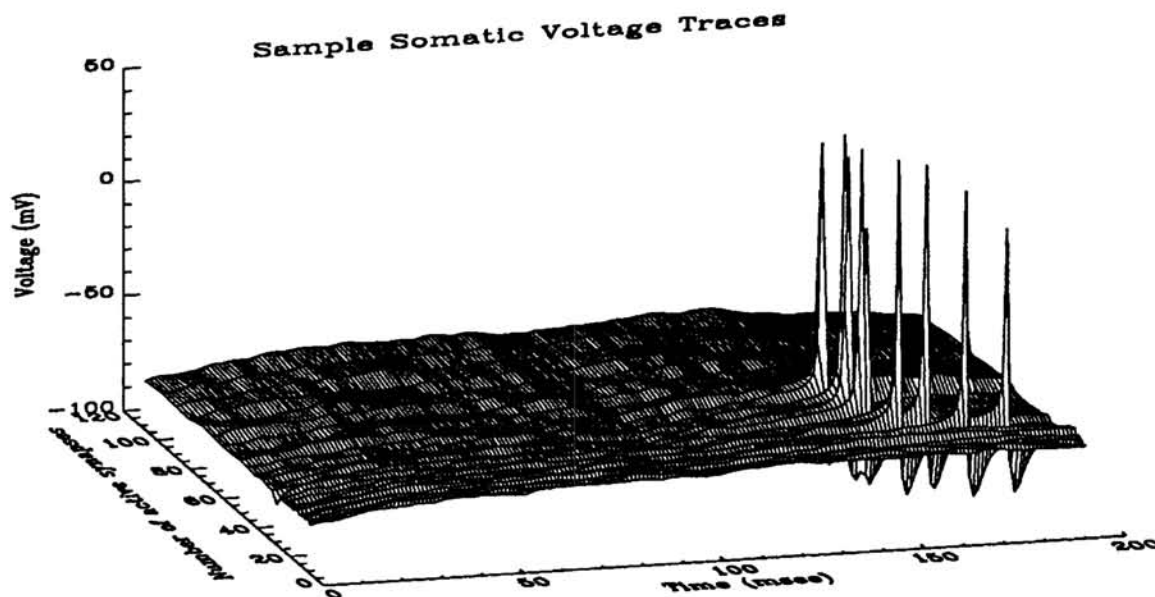

Fig. 2 Sample runs. The membrane voltage at the soma is plotted as a function of time and synaptic stimulus intensity. At $t = 100$ *msec*, a synaptic stimulus consisting of 5 pulses was activitated. The noisy baseline resulted from random synaptic input. A single action potential resulted for input intensities within a range determined by the kinetics of the cold spot.

current "threshold" to the threshold for somatic spiking. The *AHP* current required a relatively high level of activity for its activation. This *AHP* current "threshold" reflected the sigmoidal voltage dependence of N-type Ca current activation ($V_{1/2} = -28\ mV$), since only as the dendritic voltage approached $V_{1/2}$ did dendritic [$Ca^{2+}$] rise enough to activate the *AHP* current. Because $V_{1/2}$ was much higher than the threshold for somatic spiking (about $-55\ mV$ under current clamp), there was a window of stimulus strengths sufficient to trigger a somatic action potential but insufficient to activate the *AHP* current. Only within this window of between about 20 and 60 synapses (Fig. 2) did an action potential occur.

### 3.2 LOCAL NON-MONOTONIC RESPONSE FUNCTION

Because the background activity was random, the outcome of each simulation (*e.g.* Fig. 2) represented a sample of a random process. This random process can be used to define many different random variables. One variable of interest is whether a spike fired in response to a stimulus. Although this measure ignores the dynamic nature of neuronal activity, it was still relatively informative because in these simulations no more than one spike fired per experiment.

Fig. 3A shows the dependence of firing probability on stimulus strength. It was obtained by averaging over a population of simulations of the type illustrated in Fig. 2. In the absence of *AHP* current (*dotted line*), the firing probability was a sigmoidal function of activity. In its presence, the firing probability was a smooth nonmonotonic function of the activity (*solid line*). The firing probability was maximum at about 35 synapses, and occurred only in the range between about 10 and 80 synapses. The statistics illustrated in Fig. 3A quantify the nonmonotonicity that is implied by the single sample shown in Fig. 2.

Spikes required the somatic Hodgkin-Huxley-like Na and K channels. To a first approximation, the effect of these channels was to convert a continuous variable—the somatic voltage—into a discrete variable—the presence or absence of a spike. Although this approximation ignores the complex interactions between the soma and the cold spot, it is useful for a qualitative analysis. The nonmonotonic dependence of somatic activity on syn-

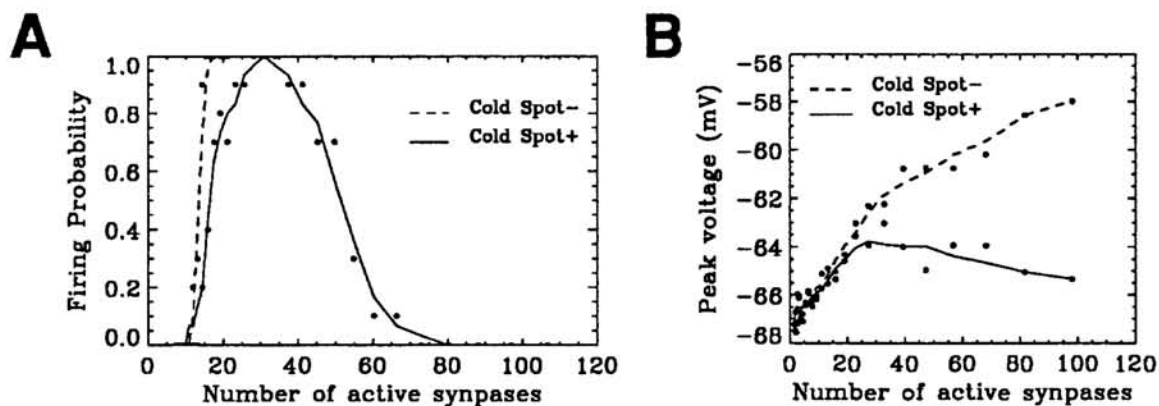

Fig. 3 Nonmonotonic input-output relation. (**A**) Each point represents the probability that at least one spike was fired at the indicated activity level. In the absence of a cold spot, the firing probability increased sharply and monotonically as the number of synapses in pathway *C/D* increased (*dotted line*). In contrast, the firing probability reached a maximum for pathway *A/B* and then decreased (*solid line*). (**B**) Each point represents the peak somatic voltage for a single simulation at the indicated activity level in the presence (*pathway A/B; solid line*) and absence (*pathway C/D; dotted line*) of a cold spot. Because each point represents the outcome of a single simulation, in contrast to the average used in *(A)*, the points reflect the variance due to the random background activity.

aptic activity was preserved even when active channels at the soma were eliminated (Fig. 3B). This result emphasizes that the critical nonlinearity was the cold spot itself.

### 3.3 NONLINEAR PATTERN SEPARATION

So far, we have treated the output as a function of a scalar—the total activity in pathway $A/B$ (or $C/D$). In Fig. 3 for example, the total activity was defined as the sum of the activities in pathway $A$ and $B$. The spatial organization of the afferents onto 2 pairs of branches—$A$ & $B$ and $C$ & $D$ (Fig. 1)—suggested considering the output as a function of the activity in the separate elements of each pair.

The effect of the cold spot can be viewed in terms of the dependence of firing as a function of separate activity in pathways $A$ and $B$ (Fig. 4). Each filled circle indicates that the neuron fired for the indicated input intensity of pathways $A$ and $B$, while a small dot indicates that it did not fire. As suggested by (Fig. 3), the firing probability was highest when the total activity in the two pathways was at some intermediate level. The neuron did not fire when the total activity in the two pathways was too large or too small. In the absence of the cold spot, only a minimum activity level was required.

In our model the probability of firing was a continuous function of the inputs. In the presence of the dendritic cold spot, the corners of this function suggested the logical operation XOR. The probability of firing was high if only one input was activated and low if both or neither was activated.

## 4  DISCUSSION

### 4.1 ASSUMPTIONS

Neuronal morphology in the present model was based on a precise reconstruction of a region CA1 pyramidal neuron. The main additional assumptions involved the kinetics and distribution of the four membrane channels, and the dynamics of $Ca^{2+}$ in the neighborhood of influx. The forms assumed for these mechanisms were biophysically plausible, and the kinetic parameters were based on estimates from a collection of experimental studies (*listed in* Lytton and Sejnowski, 1991; Zador *et al.*, 1990). Variation within the range of uncertainty of these parameters did not alter the main conclusions. The chief untested assumption of this model was the existence of cold spots. Although there is experimental evidence

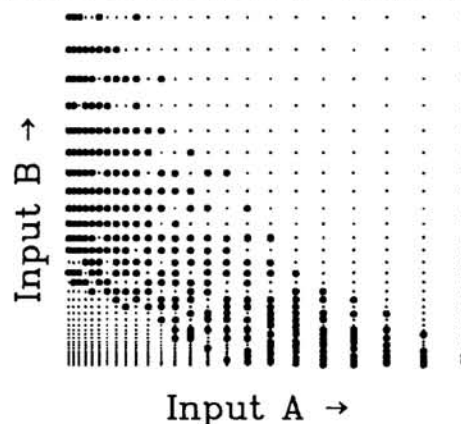

Input A →

Fig. 4 Nonlinear pattern separation  Neuronal firing is represented as a joint function of two input pathways ($A/B$). Filled circles indicate that the neuron fired for the indicated stimulus parameters. Some indication of the stochastic nature of this function, resulting form the noisy background, is given by the density of interdigitation of points and circles.

supporting the presence of both Ca and *AHP* channels in the dendrites, there is at present no direct evidence regarding their colocalization.

## 4.2   COMPUTATIONS IN SINGLE NEURONS

### 4.2.1   Neurons and Processing Elements

The limitations of the McCulloch and Pitts (1943) PE as a neuron model have long been recognized. Their threshold PE, in which the output is the weighted sum of the inputs passed through a threshold, is static, deterministic and treats all inputs equivalently. This model ignores at least three key complexities of neurons: *temporal, spatial* and *stochastic*. In subsequent years, augmented models have attempted to capture aspects of these complexities. For example, the leaky integrator (Caianiello, 1961; Hopfield, 1984) incorporates the temporal dynamics implied by the linear *RC* component of the circuit element pictured in Fig. 1B. We have demonstrated that the input-output function of a realistic neuron model can have qualitatively different behavior from that of a single processing element (PE).

### 4.2.2   Interactions Within The Dendritic Tree

The early work of Rall (1964) stressed the spatial complexity of even linear dendritic models. He noted that input from different synapses cannot be considered to arrive at a single point, the soma. Koch *et al.* (1982) extended this observation by exploring the nonlinear interactions between synaptic inputs to different regions of the dendritic tree. They emphasized that these interactions can be local in the sense that they effect subpopulations of synapses and suggested that the entire dendritic tree can be considered in terms of electrically isolated subunits. They proposed a specific role for these subunits in computing a veto—an analog AND-NOT—that might underlie directional selectivity in retinal ganglion cells. The veto was achieved through inhibitory inputs.

The underlying neuron models of Koch *et al.* (1982) and Rall (1964) were time-varying but linear, so it is not surprising that the resulting nonlinearities were monotonic. Much steeper nonlinearities were achieved by Shepherd and Brayton (1987) in a model that assumed excitable spines with fast Hodgkin-Huxley K and Na channels. These channels alone could implement the digital logic operations AND and OR. With the addition of extrinsic inhibitory inputs, they showed that a neuron could implement a full complement of digital logic operations, and concluded that a dendritic tree could in principle implement arbitrarily complex logic operations.

The emphasis of the present model differs from that of both the purely linear and of the digital approaches, although it shares their emphasis on the locality of dendritic computation. Because the cold spot involved strongly nonlinear channels, it implemented a nonmonotonic response function, in contrast to strictly linear dendritic models. At the same time, the present model retained the essentially analog nature of intraneuronal signalling, in contrast to the digital dendritic models. This analog mode seems better suited to processing large numbers of noisy inputs because it preserves the uncertainties rather than making an immediate decision. Focussing on the analog nature of the response eliminated the requirement for operating within the digital range of channel dynamics.

The NMDA receptor-gated channel can give rise to an analog AND with a weaker voltage-dependence than that induced by fast Na and K channels. Mel (1992) described a model in which synapses mediating increases to both the NMDA and AMPA conductances were distributed across the dendritic tree of a cortical neuron. When the synaptic activity was dis-

tributed in appropriately sized clusters, the resulting neuronal response function was reminiscent of that of a sigma-pi unit. With suitable preprocessing of inputs, the neuron could perform complex pattern discrimination.

A unique feature of the present model is that functional inhibition arose from purely excitatory inputs. This mechanism underlying this inhibition —the AHP current—was intrinsic to the membrane. In both the Koch *et al.* (1982) and Brayton and Shepherd (1987) models, the veto or NOT operation was achieved through extrinsic synaptic inhibition. This requires additional neuronal circuitry. In the case of a dedicated sensory system like the directionally selective retinal granule cell, it is not unreasonable to imagine that the requisite neuronal circuitry is hardwired. In the limiting case of the digital model, the requisite circuitry would involve a separate inhibitory interneuron for each NOT-gate.

### 4.2.3 Adaptive Dendritic Computation

What algorithms can harness the computational potential of the dendritic tree? Adaptive dendritic computation is a very new subject. Brown *et al.* (1991, 1992) developed a model in which synapses distributed across the dendritic tree showed interesting forms of spatial self-organization. Synaptic plasticity was governed by a local biophysically-motivated Hebb rule (Zador *et al.*, 1990). When temporally correlated but spatially uncorrelated inputs were presented to the neuron, spatial clusters of strengthened synapses emerged within the dendritic tree. The neuron converted a temporal correlation into a spatial correlation.

The computational role of clusters of strengthened synapses within the dendritic tree becomes important in the presence of nonlinear membrane. If the dendrites are purely passive, then saturation ensures that the current injected per synapse actually *decreases* as the clustering increases. If purely regenerative nonlinearities are present (Brayton and Shepherd, 1987; Mel, 1992), then the response increases. The cold spot extends the range of local dendritic computations.

What might control the formation and distribution of the cold spot itself? Cold spots might arise from the fortuitous colocalization of Ca and $K_{AHP}$ channels. Another possibility is that some specific biophysical mechanism creates cold spots in a use-dependent manner. Candidate mechanisms might involve local changes in second messengers such as $[Ca^{2+}]$ or longitudinal potential gradients (*cf.* Poo, 1985). Bell (1992) has shown that this second mechanism can induce computationally interesting distributions of membrane channels.

### 4.3 WHY STUDY SINGLE NEURONS?

We have illustrated an important functional difference between a single neuron and a PE. A neuron with cold spots can perform extensive local processing in the dendritic tree, giving rise to a complex mapping between input and output. A neuron may perhaps be likened to a "micronet" of simpler PEs, since any mapping can be approximated by a sufficiently complex network of sigmoidal units (Cybenko, 1989). This raises the objection that since micronets represent just a subset of all neural networks, there may be little to be gained by studying the properties of the special case of neurons.

The intuitive justification for studying single neurons is that they represent a large but highly constrained subset that may have very special properties. Knowledge of the properties general to all sufficiently complex PE networks may provide little insight into the properties specific to single neurons. These properties may have implications for the behavior of circuits of neurons. It is not unreasonable to suppose that adaptive mechanisms in biological circuits will utilize the specific strengths of single neurons.

## Acknowledgments

We thank Michael Hines for providing NEURON-MODL assisting with new membrane mechanisms. This research was supported by grants from the Office of Naval Research, the Defense Advanced Research Projects Agency, and the Air Force Office of Scientific Research.

## References

Bell, T. (1992) *Neural information processing systems* 4 *(in press)*.

Borg-Graham, L.J. (1991) In H. Wheal and J. Chad (Eds.) *Cellular and Molecular Neurobiology: A Practical Approach*. New York: Oxford University Press.

Brown, T.H. and Zador, A.M. (1990). In G. Shepherd (Ed.) *The synaptic organization of the brain* (Vol. 3, pp. 346-388). New York: Oxford University Press.

Brown, T.H., Mainen, Z.F., Zador, A.M. and Claiborne, B.J. (1991) *Neural information processing systems* 3: 39-45.

Brown, T.H., Zador, A.M., Mainen, Z.F., and Claiborne, B.J. (1992). In: *Single neuron computation*. Eds. T. McKenna, J. Davis, and S.F. Zornetzer. Academic Press *(in press)*.

Caianiello, E.R. (1961) *J. Theor. Biol.* 1: 209-235.

Claiborne, B.J., Zador, A.M., Mainen, Z.F., and Brown, T.H. (1992). In: *Single neuron computation*. Eds. T. McKenna, J. Davis, and S.F. Zornetzer. Academic Press *(in press)*.

Cybenko, G. (1989) *Math. Control, Signals Syst.* 2: 303-314.

Hines, M. (1989). *Int. J. Biomed. Comp*, 24: 55-68.

Hodgkin, A.L. and Huxley, A.F. (1952) *J. Physiol.* 117: 500-544.

Hopfield, J.J. (1984) *Proc. Natl. Acad. Sci. USA* 81: 3088-3092.

Jack, J. Noble, A. and Tsien, R.W. (1975) *Electrical current flow in excitable membranes*. London: Oxford Press.

Jones, O.T., Kunze, D.L. and Angelides, K.J. (1989) *Science.* 244:1189-1193.

Koch, C., Poggio, T. and Torre, V. (1982) *Proc. R. Soc. London B.* 298: 227-264.

Lancaster, B. and Nicoll, R.A. (1987) *J. Physiol.* 389: 187-203.

Lancaster, B., Perkel, D.J., and Nicoll, R.A. (1991) *J. Neurosci.* 11:23-30.

Lytton, W.W. and Sejnowski, T.J. (1991) *J. Neurophys.* 66: 1059-1079.

McCulloch, W.S. and Pitts, W. (1943) *Bull. Math. Biophys.* 5: 115-137.

Mel, B. (1992) *Neural Computation (in press)*.

Poo, M-m. (1985) *Ann. Rev. Neurosci.* 8: 369-406.

Rall, W. (1977) In: *Handbook of physiology*. Eds. E. Kandel and S. Geiger. Washington D.C.: American Physiological Society, pp. 39-97.

Rall, W. (1964) In: *Neural theory and modeling*. Ed. R.F. Reiss. Stanford Univ. Press, pp. 73-79.

Shepherd, G.M. and Brayton, R.K. (1987) *Neuroscience* 21: 151-166.

Zador, A., Koch, C. and Brown, T.H. (1990) *Proc. Natl. Acad. Sci. USA* 87: 6718-6722.
